# The Manhattan World Assumption: Regularities in scene statistics which enable Bayesian inference

James M. Coughlan
Smith-Kettlewell Eye Research Inst.
2318 Fillmore St.
San Francisco, CA 94115
*coughlan@ski.org*

A.L. Yuille
Smith-Kettlewell Eye Research Inst.
2318 Fillmore St.
San Francisco, CA 94115
*yuille@ski.org*

## Abstract

Preliminary work by the authors made use of the so-called "Manhattan world" assumption about the scene statistics of city and indoor scenes. This assumption stated that such scenes were built on a cartesian grid which led to regularities in the image edge gradient statistics. In this paper we explore the general applicability of this assumption and show that, surprisingly, it holds in a large variety of less structured environments including rural scenes. This enables us, from a single image, to determine the orientation of the viewer relative to the scene structure and also to detect target objects which are not aligned with the grid. These inferences are performed using a Bayesian model with probability distributions (e.g. on the image gradient statistics) learnt from real data.

## 1 Introduction

In recent years, there has been growing interest in the statistics of natural images (see Huang and Mumford [4] for a recent review). Our focus, however, is on the discovery of scene statistics which are useful for solving visual inference problems. For example, in related work [5] we have analyzed the statistics of filter responses on and off edges and hence derived effective edge detectors.

In this paper we present results on statistical regularities of the image gradient responses as a function of the global scene structure. This builds on preliminary work [2] on city and indoor scenes. This work observed that such scenes are based on a cartesian coordinate system which puts (probabilistic) constraints on the image gradient statistics.

Our current work shows that this so-called "Manhattan world" assumption about the scene statistics applies far more generally than urban scenes. Many rural scenes contain sufficient structure on the distribution of edges to provide a natural cartesian reference frame for the viewer. The viewers' orientation relative to this frame can be determined by Bayesian inference. In addition, certain structures in the scene stand out by being unaligned to this natural reference frame. In our theory such

structures appear as "outlier" edges which makes it easier to detect them. Informal evidence that human observers use a form of the Manhattan world assumption is provided by the Ames room illusion, see figure (6), where the observers appear to erroneously make this assumption, thereby grotesquely distorting the sizes of objects in the room.

## 2 Previous Work and Three- Dimensional Geometry

Our preliminary work on city scenes was presented in [2]. There is related work in computer vision for the detection of vanishing points in 3-d scenes [1], [6] (which proceeds through the stages of edge detection, grouping by Hough transforms, and finally the estimation of the geometry).

We refer the reader to [3] for details on the geometry of the Manhattan world and report only the main results here. Briefly, we calculate expressions for the orientations of $x, y, z$ lines imaged under perspective projection in terms of the orientation of the camera relative to the $x, y, z$ axes. The camera orientation relative to the $xyz$ axis system may be specified by three Euler angles: the *azimuth* (or *compass angle*) $\alpha$, corresponding to rotation about the $z$ axis, the *elevation* $\beta$ above the $xy$ plane, and the *twist* $\gamma$ about the camera's line of sight. We use $\vec{\Psi} = (\alpha, \beta, \gamma)$ to denote all three Euler angles of the camera orientation. Our previous work [2] assumed that the elevation and twist were both zero which turned out to be invalid for many of the images presented in this paper.

We can then compute the normal orientation of lines parallel to the $x, y, z$ axes, measured in the image plane, as a function of film coordinates $(u, v)$ and the camera orientation $\vec{\Psi}$. We express the results in terms of orthogonal unit camera axes $\vec{a}, \vec{b}$ and $\vec{c}$, which are aligned to the body of the camera and are determined by $\vec{\Psi}$. For $x$ lines (see Figure 1, left panel) we have $\tan \theta_x = -(uc_x + fa_x)/(vc_x + fb_x)$, where $\theta_x$ is the normal orientation of the $x$ line at film coordinates $(u, v)$ and $f$ is the focal length of the camera. Similarly, $\tan \theta_y = -(uc_y + fa_y)/(vc_y + fb_y)$ for $y$ lines and $\tan \theta_z = -(uc_z + fa_z)/(vc_z + fb_z)$ for $z$ lines. In the next section will see how to relate the normal orientation of an object boundary (such as $x, y, z$ lines) at a point $(u, v)$ to the magnitude and direction of the image gradient at that location.

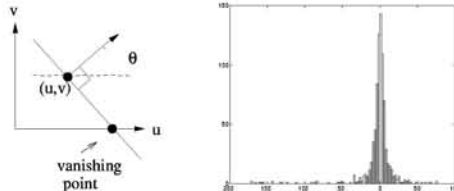

Figure 1: (Left) Geometry of an $x$ line projected onto $(u, v)$ image plane. $\theta$ is the normal orientation of the line in the image. (Right) Histogram of edge orientation error (displayed modulo 180°). Observe the strong peak at 0°, indicating that the image gradient direction at an edge is usually very close to the true normal orientation of the edge.

## 3 $P_{on}$ and $P_{off}$: Characterizing Edges Statistically

Since we do not know where the $x, y, z$ lines are in the image, we have to infer their locations and orientations from image gradient information. This inference is done

using a *purely local* statistical model of edges. A key element of our approach is that it allows the model to infer camera orientation without having to group pixels into $x, y, z$ lines. Most grouping procedures rely on the use of binary edge maps which often make premature decisions based on too little information. The poor quality of some of the images – underexposed and overexposed – makes edge detection particularly difficult, as well as the fact that some of the images lack $x, y, z$ lines that are long enough to group reliably.

Following work by Konishi *et al* [5], we determine probabilities $P_{on}(E_{\vec{u}})$ and $P_{off}(E_{\vec{u}})$ for the probabilities of the image gradient magnitude $E_{\vec{u}}$ at position $\vec{u}$ in the image *conditioned on whether we are on or off an edge*. These distributions quantify the tendency for the image gradient to be high on object boundaries and low off them, see Figure 2. They were learned by Konishi *et al* for the Sowerby image database which contains one hundred presegmented images.

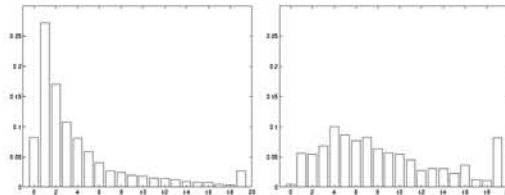

Figure 2: $P_{off}(y)$ (left) and $P_{on}(y)$(right), the empirical histograms of edge responses off and on edges, respectively. Here the response $y = \left|\vec{\nabla} I\right|$ is quantized to take 20 values and is shown on the horizontal axis. Note that the peak of $P_{off}(y)$ occurs at a lower edge response than the peak of $P_{on}(y)$.

We extend the work of Konishi *et al* by putting probability distributions on how accurately the image gradient *direction* estimates the true normal direction of the edge. These were learned for this dataset by measuring the true orientations of the edges and comparing them to those estimated from the image gradients.

This gives us distributions on the magnitude and direction of the intensity gradient $P_{on}(\vec{E}_{\vec{u}}|\theta), P_{off}(\vec{E}_{\vec{u}})$, where $\vec{E}_{\vec{u}} = (E_{\vec{u}}, \phi_{\vec{u}})$, $\theta$ is the true normal orientation of the edge, and $\phi_{\vec{u}}$ is the gradient direction measured at point $\vec{u} = (u, v)$. We make a *factorization assumption* that $P_{on}(\vec{E}_{\vec{u}}|\theta) = P_{on}(E_{\vec{u}})P_{ang}(\phi_{\vec{u}} - \theta)$ and $P_{off}(\vec{E}_{\vec{u}}) = P_{off}(E_{\vec{u}})U(\phi_{\vec{u}})$. $P_{ang}(.)$ (with argument evaluated modulo $2\pi$ and normalized to 1 over the range 0 to $2\pi$) is based on experimental data, see Figure 1 (right), and is peaked about 0 and $\pi$. In practice, we use a simple box-shaped function to model the distribution: $P_{ang}(\delta\theta) = (1 - \epsilon)/4\tau$ if $\delta\theta$ is within angle $\tau$ of 0 or $\pi$, and $\epsilon/(2\pi - 4\tau)$ otherwise (i.e. the chance of an angular error greater than $\pm\tau$ is $\epsilon$ ). In our experiments $\epsilon = 0.1$ and $\tau = 4°$ for indoors and $6°$ outdoors. By contrast, $U(.) = 1/2\pi$ is the uniform distribution.

## 4   Bayesian Model

We devised a Bayesian model which combines knowledge of the three-dimensional geometry of the Manhattan world with statistical knowledge of edges in images. The model assumes that, while the majority of pixels in the image convey no information about camera orientation, most of the pixels with high edge responses arise from the presence of $x, y, z$ lines in the three-dimensional scene. An important feature of the Bayesian model is that *it does not force us to decide prematurely which pixels*

*are on and off an object boundary* (or whether an on pixel is due to $x, y,$ or $z$), *but allows us to sum over all possible interpretations of each pixel.*

The image data $\vec{E}_{\vec{u}}$ at a single pixel $\vec{u}$ is explained by one of five models $m_{\vec{u}}$: $m_{\vec{u}} = 1, 2, 3$ mean the data is generated by an edge due to an $x, y, z$ line, respectively, in the scene; $m_{\vec{u}} = 4$ means the data is generated by an outlier edge (not due to an $x, y, z$ line); and $m_{\vec{u}} = 5$ means the pixel is off-edge. The prior probability $P(m_{\vec{u}})$ of each of the edge models was estimated empirically to be $0.02, 0.02, 0.02, 0.04, 0.9$ for $m_{\vec{u}} = 1, 2, \ldots, 5$.

Using the factorization assumption mentioned before, we assume the probability of the image data $\vec{E}_{\vec{u}}$ has two factors, one for the magnitude of the edge strength and another for the edge direction:

$$P(\vec{E}_{\vec{u}}|m_{\vec{u}}, \vec{\Psi}, \vec{u}) = P(E_{\vec{u}}|m_{\vec{u}})P(\phi_{\vec{u}}|m_{\vec{u}}, \vec{\Psi}, \vec{u}) \qquad (1)$$

where $P(E_{\vec{u}}|m_{\vec{u}})$ equals $P_{off}(E_{\vec{u}})$ if $m_{\vec{u}} = 5$ or $P_{on}(E_{\vec{u}})$ if $m_{\vec{u}} \neq 5$. Also, $P(\phi_{\vec{u}}|m_{\vec{u}}, \vec{\Psi}, \vec{u})$ equals $P_{ang}(\phi_{\vec{u}} - \theta(\vec{\Psi}, m_{\vec{u}}, \vec{u}))$ if $m_{\vec{u}} = 1, 2, 3$ or $U(\phi_{\vec{u}})$ if $m_{\vec{u}} = 4, 5$. Here $\theta(\vec{\Psi}, m_{\vec{u}}, \vec{u}))$ is the predicted normal orientation of lines determined by the equation $\tan\theta_x = -(uc_x + fa_x)/(vc_x + fb_x)$ for x lines, $\tan\theta_y = -(uc_y + fa_y)/(vc_y + fb_y)$ for y lines, and $\tan\theta_z = -(uc_z + fa_z)/(vc_z + fb_z)$ for z lines.

In summary, the edge strength probability is modeled by $P_{on}$ for models 1 through 4 and by $P_{off}$ for model 5. For models 1,2 and 3 the edge orientation is modeled by a distribution which is peaked about the appropriate orientation of an $x, y, z$ line predicted by the camera orientation at pixel location $\vec{u}$; for models 4 and 5 the edge orientation is assumed to be uniformly distributed from 0 through $2\pi$.

Rather than decide on a particular model at each pixel, we marginalize over all five possible models (i.e. creating a mixture model):

$$P(\vec{E}_{\vec{u}}|\vec{\Psi}, \vec{u}) = \sum_{m_{\vec{u}}=1}^{5} P(\vec{E}_{\vec{u}}|m_{\vec{u}}, \vec{\Psi}, \vec{u})P(m_{\vec{u}}) \qquad (2)$$

Now, to combine evidence over all pixels in the image, denoted by $\{\vec{E}_{\vec{u}}\}$, we assume that the image data is conditionally independent across all pixels, given the camera orientation $\vec{\Psi}$:

$$P(\{\vec{E}_{\vec{u}}\}|\vec{\Psi}) = \prod_{\vec{u}} P(\vec{E}_{\vec{u}}|\vec{\Psi}, \vec{u}) \qquad (3)$$

(Although the conditional independence assumption neglects the coupling of gradients at neighboring pixels, it is a useful approximation that makes the model computationally tractable.) Thus the posterior distribution on the camera orientation is given by $\prod_{\vec{u}} P(\vec{E}_{\vec{u}}|\vec{\Psi}, \vec{u})P(\vec{\Psi})/Z$ where $Z$ is a normalization factor and $P(\vec{\Psi})$ is a uniform prior on the camera orientation.

To find the MAP (maximum a posterior) estimate, our algorithm maximizes the log posterior term $\log[P(\{\vec{E}_{\vec{u}}\}|\vec{\Psi})P(\vec{\Psi})] = \log P(\vec{\Psi}) + \sum_{\vec{u}} \log[\sum_{m_{\vec{u}}} P(\vec{E}_{\vec{u}}|m_{\vec{u}}, \vec{\Psi}, \vec{u})P(m_{\vec{u}})]$ numerically by searching over a quantized set of compass directions $\vec{\Psi}$ in a certain range. For details on this procedure, as well as coarse-to-fine techniques for speeding up the search, see [3].

# 5 Experimental Results

This section presents results on the domains for which the viewer orientation relative to the scene can be detected using the Manhattan world assumption. In particular, we demonstrate results for: (I) indoor and outdoor scenes (as reported in [2]), (II) rural English road scenes, (III) rural English fields, (IV) a painting of the French countryside, (V) a field of broccoli in the American mid-west, (VI) the Ames room, and (VII) ruins of the Parthenon (in Athens). The results show strong success for inference using the Manhattan world assumption even for domains in which it might seem unlikely to apply. (Some examples of failure are given in [3]. For example, a helicopter in a hilly scene where the algorithm mistakenly interprets the hill silhouettes as horizontal lines).

The first set of images were of city and indoor scenes in San Francisco with images taken by the second author [2]. We include four typical results, see figure 3, for comparison with the results on other domains.

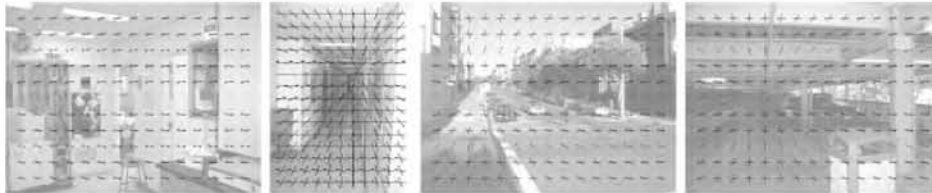

Figure 3: Estimates of the camera orientation obtained by our algorithm for two indoor scenes (left) and two outdoor scenes (right). The estimated orientations of the $x, y$ lines, derived for the estimated camera orientation $\vec{\Psi}$, are indicated by the black line segments drawn on the input image. (The $z$ line orientations have been omitted for clarity.) At each point on a subgrid two such segments are drawn – one for $x$ and one for $y$. In the image on the far left, observe how the $x$ directions align with the wall on the right hand side and with features parallel to this wall. The $y$ lines align with the wall on the left (and objects parallel to it).

We now extend this work to less structured scenes in the English countryside. Figure (4) shows two images of roads in rural scenes and two fields. These images come from the Sowerby database. The next three images were either downloaded from the web or digitized (the painting). These are the mid-west broccoli field, the Parthenon ruins, and the painting of the French countryside.

# 6 Detecting Objects in Manhattan world

We now consider applying the Manhattan assumption to the alternative problem of detecting target objects in background clutter. To perform such a task effectively requires modelling the properties of the background clutter in addition to those of the target object. It has recently been appreciated that good statistical modelling of the image background can improve the performance of target recognition [7].

The Manhattan world assumption gives an alternative way of probabilistically modelling background clutter. The background clutter will correspond to the regular structure of buildings and roads and its edges will be aligned to the Manhattan grid. The target object, however, is assumed to be unaligned (at least, in part) to this grid. *Therefore many of the edges of the target object will be assigned to model 4 by the algorithm.* (Note the algorithm first finds the MAP estimate $\vec{\Psi}^*$ of the

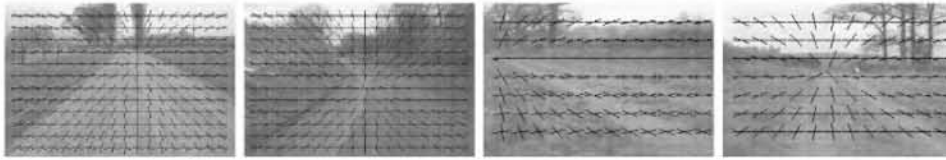

Figure 4: Results on rural images in England without strong Manhattan structure. Same conventions as before. Two images of roads in the countryside (left panels) and two images of fields (right panel).

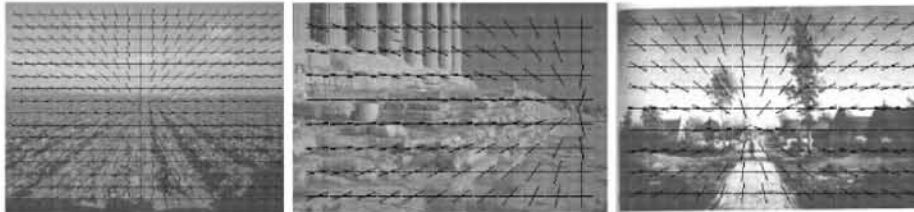

Figure 5: Results on an American mid-west broccoli field, the ruins of the Parthenon, and a digitized painting of the French countryside.

compass orientation, see section (4), and then estimates the model by doing MAP of $P(m_{\vec{u}}|\vec{E}_{\vec{u}}, \vec{\Psi}^*, \vec{u})$ to estimate $m_{\vec{u}}$ for each pixel $\vec{u}$.) This enables us to significantly simplify the detection task by removing all edges in the images except those assigned to model 4.

The Ames room, see figure (6), is a geometrically distorted room which is constructed so as to give the false impression that it is built on a cartesian coordinate frame when viewed from a special viewpoint. Human observers assume that the room is indeed cartesian despite all other visual cues to the contrary. This distorts the apparent size of objects so that, for example, humans in different parts of the room appear to have very different sizes. In fact, a human walking across the room will appear to change size dramatically. Our algorithm, like human observers, interprets the room as being cartesian and helps identify the humans in the room as outlier edges which are unaligned to the cartesian reference system.

# 7   Summary and Conclusions

We have demonstrated that the Manhattan world assumption applies to a range of images, rural and otherwise, in addition to urban scenes. We demonstrated a Bayesian model which used this assumption to infer the orientation of the viewer relative to this reference frame and which could also detect outlier edges which are unaligned to the reference frame. A key element of this approach is the use of image gradient statistics, learned from image datasets, which quantify the distribution of the image gradient magnitude and direction on and off object boundaries. We expect that there are many further image regularities of this type which can be used for building effective artificial vision systems and which are possibly made use of by biological vision systems.

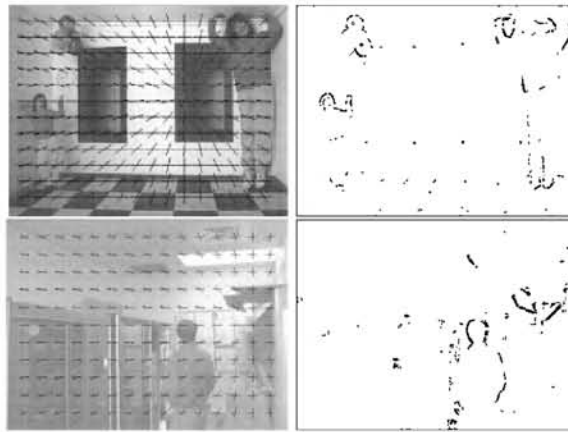

Figure 6: Detecting people in Manhattan world. The left images (top and bottom) show the estimated scene structure. The right images show that people stand out as residual edges which are unaligned to the Manhattan grid. The Ames room (top panel) *violates* the Manhattan assumption but human observers, and our algorithm, interpret it as if it satisfied the assumptions. In fact, despite appearances, the two people in the Ames room are really the same size.

## Acknowledgments

We want to acknowledge funding from NSF with award number IRI-9700446, support from the Smith-Kettlewell core grant, and from the Center for Imaging Sciences with Army grant ARO DAAH049510494. This work was also supported by the National Institute of Health (NEI) with grant number RO1-EY 12691-01. It is a pleasure to acknowledge email conversations with Song Chun Zhu about scene clutter. We gratefully acknowledge the use of the Sowerby image dataset from Sowerby Research Centre, British Aerospace.

## References

[1] B. Brillault-O'Mahony. "New Method for Vanishing Point Detection". *Computer Vision, Graphics, and Image Processing.* 54(2). pp 289-300. 1991.

[2] J. Coughlan and A.L. Yuille. "Manhattan World: Compass Direction from a Single Image by Bayesian Inference". *Proceedings International Conference on Computer Vision* ICCV'99. Corfu, Greece. 1999.

[3] J. Coughlan and A.L. Yuille. "Manhattan World: Orientation and Outlier Detection by Bayesian Inference." Submitted to International Journal of Computer Vision. 2000.

[4] J. Huang and D. Mumford. "Statistics of Natural Images and Models". In *Proceedings Computer Vision and Pattern Recognition* CVPR'99. Fort Collins, Colorado. 1999.

[5] S. Konishi, A. L. Yuille, J. M. Coughlan, and S. C. Zhu. "Fundamental Bounds on Edge Detection: An Information Theoretic Evaluation of Different Edge Cues." *Proc. Int'l conf. on Computer Vision and Pattern Recognition*, 1999.

[6] E. Lutton, H. Maître, and J. Lopez-Krahe. "Contribution to the determination of vanishing points using Hough transform". *IEEE Trans. on Pattern Analysis and Machine Intelligence.* 16(4). pp 430-438. 1994.

[7] S. C. Zhu, A. Lanterman, and M. I. Miller. "Clutter Modeling and Performance Analysis in Automatic Target Recognition". In *Proceedings Workshop on Detection and Classification of Difficult Targets.* Redstone Arsenal, Alabama. 1998.
